# Beyond Categories: The Visual Memex Model for Reasoning About Object Relationships

**Tomasz Malisiewicz,  Alexei A. Efros**
Robotics Institute
Carnegie Mellon University
`{tmalisie,efros}@cs.cmu.edu`

## Abstract

The use of context is critical for scene understanding in computer vision, where the recognition of an object is driven by both local appearance and the object's relationship to other elements of the scene (context). Most current approaches rely on modeling the relationships between object categories as a source of context. In this paper we seek to move beyond categories to provide a richer appearance-based model of context. We present an exemplar-based model of objects and their relationships, the *Visual Memex*, that encodes both local appearance and 2D spatial context between object *instances*. We evaluate our model on Torralba's proposed Context Challenge against a baseline category-based system. Our experiments suggest that moving beyond categories for context modeling appears to be quite beneficial, and may be the critical missing ingredient in scene understanding systems.

## 1  Introduction

Image understanding is one of the Holy Grail problems in computer vision. Understanding a scene arguably requires parsing the image into its constituent objects. In real scenes composed of many different objects, the spatial configuration of one object can facilitate recognition of related objects [1], and quite often ambiguities in recognition cannot be resolved without looking beyond the spatial extent of the object in question. Thus, algorithms which jointly recognize many objects at once by taking account of contextual relationships have been quite popular. While early systems relied on hand-coded rules for inter-object context (e.g. [2, 3]), more modern approaches typically perform inference in a probabilistic graphical model with respect to categories where object interactions are modeled as higher order potentials [4, 5, 6, 7, 8, 9, 10]. One important implicit assumption made by all such models is that interactions between object *instances* can be adequately modeled as relationships between human-defined object *categories*.

In this paper we challenge this "category assumption" for object-object interactions and propose a novel category-free approach for modeling object relationships. We propose a new framework, the *Visual Memex Model*, for representing and reasoning about object identities and their contextual relationships in an exemplar-based, non-parametric way. We evaluate our model on Antonio Torralba's proposed Context Challenge [11] against a baseline category-based system.

## 2  Motivation

The use of categories (classes) to represent concepts (e.g. visual objects) is so prevalent in computer vision and machine learning that most researchers don't give it a second thought. Faced with a new task, one simply carves up the solution space into classes (e.g. cars, people, buildings), assigns class labels to training examples and applies one of the many popular classifiers to arrive at a solution.

However, we believe that it is worthwhile to re-examine the basic assumption behind categorization, and especially its role in modeling relationships between objects.

Theories of categorization date back to the ancient Greeks. Aristotle defined categories as discrete entities characterized by a set of properties shared by all their members [12]. His categories are mutually exclusive, and every member of a category is equal. This classical view is still the most widely accepted way of reasoning about categories and taxonomies in hard sciences. However, as pointed out by Wittgenstein, this is almost certainly not the way most of our everyday concepts work (e.g. what is the set of properties that define the concept "game" and nothing else? [13]). Empirical evidence for typicality (e.g. a robin is a more commonly cited example of "bird" than a chicken) and multiple category memberships (e.g. chicken is both "bird" and "food") further complicate the Aristotelian view.

The ground-breaking work of cognitive psychologist Eleanor Rosch [14] demonstrated that humans do not cut up the world into neat categories defined by shared properties, but instead use *similarity* as the basis of categorization. Her Prototype Theory postulates that an object's class is determined by its similarity to (a set of) prototypes which define each category, allowing for varying degree of membership. Such Prototype models have been successfully used for object recognition [15, 16]. Going even further, Exemplar Theory [17, 18] rejects the need for explicit category representation, arguing instead that a concept can be implicitly formed via all its observed instances. This allows for a dynamic definition of categories based on data availability and task (e.g. an object can be a vehicle, a car, a Volvo, or Bob's Volvo). A recent operationalization of the exemplar model in the visual domain can be found in [19].

But it might not be too productive to concentrate on the various categorization theories without considering the final aim – what do we need categories for? One argument is that categorization is a tool to facilitate knowledge transfer. E.g. having been attacked once by a tiger, it's critically important to determine if a newly observed object belongs to the tiger category so as to utilize the information from the previous encounter. Note that here recognizing the explicit category is unimportant, as long as the two tigers could be associated with each other. Guided by this intuition and evidence from cognitive neuroscience, Bar [20] outlined the importance of analogies, associations, and prediction in the human brain. He argues that the goal of visual perception is not to recognize an object in the traditional sense of categorizing it (i.e. asking 'what is this?'), but instead linking the input with an analogous representation in memory (i.e. asking 'what is this *like*?'). Once a novel input is linked with analogous representations, associated representations are activated rapidly and predict the representations of what is most likely to occur next.

These ideas regarding analogies, associations, and prediction are surprisingly similar to Vannevar Bush's 1945 concept of the *Memex* [21] – which was seen decades later as pioneering hypertext and the World Wide Web. Concerned with the transmission and accessibility of scientific ideas, Bush faulted the "artificiality of systems of indexing" and proposed the *Memory Extender (Memex)*, a physical device which would help find information based on association instead of strict categorical indexing. The associative links were to be entered manually by the user and could be of several different types. Chains of links would form into longer "associative trails" creating new narratives in the concept space. For Bush "the process of tying two items together is the important thing."

Inspired by these diverse ideas that are, nonetheless, all pointing in the same general direction, we have been motivated to try to evaluate them on a concrete problem, to see if they can offer benefits over the more traditional classification framework. One particular area where we feel these ideas might prove very useful is in modeling relationships between objects within an image. Therefore, in this paper we propose, in an homage to Bush, the *Visual Memex Model*, as a first step towards operationalizing the direct modeling of associations between visual objects, and compare it with more standard tools for the same task.

## 3 The Visual Memex Model

Our starting point is Vannevar Bush's observation that strict categorical indexing of concepts has severe limitations. Abandoning rigid object categories, we embrace Bush's and Bar's belief in the primary role of associations, but unlike Bush, we aim to discover these associations automatically from the data. At the core of our model is an exemplar-based representation of objects [18, 19]. The Visual Memex can then be thought of as a vast graph, with nodes representing all the object instances

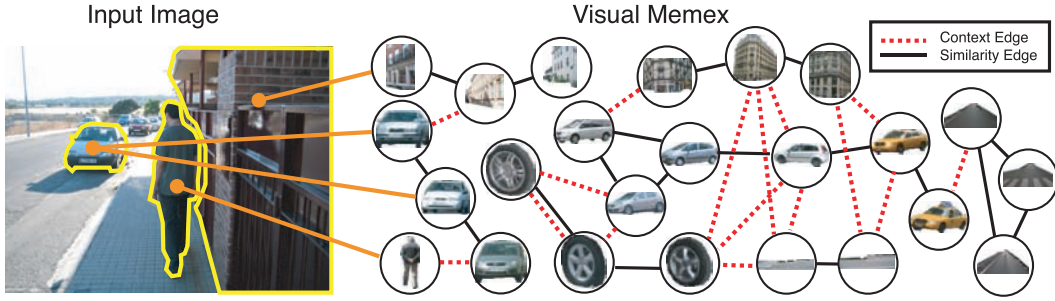

Figure 1: The **Visual Memex** graph encodes object similarity (solid black edge) and spatial context (dotted red edge) between pairs of object exemplars. A spatial context feature is stored for each context edge. The Memex graph can be used to interpret a new image (left) by associating image segments with exemplars in the graph (orange edges) and propagating the information. *Figure best viewed in color.*

in the dataset, and arcs representing the different types of associations between them (Figure 1). There are two types of arcs in our model, encoding two different relationships between objects: 1) visual similarity (e.g. this car looks like that car), and 2) contextual associations (e.g. this car is next to this building).

Once the graph is built, it can be used to interpret a novel image (Figure 1, left) by first connecting segments within the image with similar stored exemplars, and then propagating contextual information between these exemplars through the graph. When an exemplar gets activated, visually similar exemplars as well as other contextually relevant objects get activated as well. This way, exemplar-to-exemplar similarity in the Memex graph can serve as Bush's "trails" to link concepts together in a non-parametric, query-dependent way, without the use of predefined categories. For example, in Figure 1, we should be able to infer that a car seen from the rear often co-occurs with an oblique building wall (but not a frontal wall) – something which category-based models would be hard-pressed to achieve.

Formally, we define the Visual Memex Model as a graph $G = (V, E_S, E_C, \{D\}, \{\mathbf{f}\})$ consisting of $N$ object exemplar nodes $V$, similarity edges $E_S$, context edges $E_C$, $N$ per-exemplar similarity functions $\{D\}$, and the spatial features $\{\mathbf{f}\}$ associated with each context edge. We now describe how to learn the similarity functions $\{D\}$ from data to create the structure of the Visual Memex.

## 3.1 Similarity Edges

We use the per-exemplar distance-function learning algorithm of Malisiewicz *et al* [19] to learn the object similarity edges. For each exemplar, the algorithm learns which other exemplars it is similar to as well as a distance function. A distance function is a linear combination of elementary distances used to measure similarity to the exemplar. We use the same 14 color, shape, texture, and location features as used in [19]. For the $j$-th exemplar, $\mathbf{w}_j$ is the vector of 14 weights, $b_j$ is a scalar bias, and $\boldsymbol{\alpha}_j \in \{0, 1\}^{|C|}$ is a binary indicator vector which encodes which other exemplars the current exemplar is similar to. We solve $[\mathbf{w}_j^*, b_j^*, \boldsymbol{\alpha}_j^*] = \arg\min_{\mathbf{w}, b, \boldsymbol{\alpha}} f_j(\mathbf{w}, b, \boldsymbol{\alpha})$, but since the exemplars' optimization problems are independent we drop the $j$ suffix for clarity. Let $\mathbf{d}_i$ be the vector of 14 Euclidean distances between the exemplar whose similarity we are learning (the focal exemplar) and the $i$-th exemplar. $C$ is the set of exemplars that have the same label as the focal exemplar. Let $L(x) = max(1 - x, 0)^2$ be the hinge-squared loss function. A different $\mathbf{w}$, $b$, and $\boldsymbol{\alpha}$ are learned per-exemplar by optimizing the following functional:

$$f(\mathbf{w}, b, \boldsymbol{\alpha}) = \frac{\lambda}{2}||\mathbf{w}||^2 + \sum_{i \in C} \boldsymbol{\alpha}_i L(-(\mathbf{w}^T \mathbf{d}_i + b)) + \sum_{i \notin C} L(\mathbf{w}^T \mathbf{d}_i + b) - \sigma||\boldsymbol{\alpha}||^2 \qquad (1)$$

We minimize the above SVM-like objective function via an alternating optimization strategy as in [19]. The algorithm uses labels (see Section 3.3) during learning where the regularization term favors connecting to many similarly-labeled exemplars and the loss term favors separability in distance

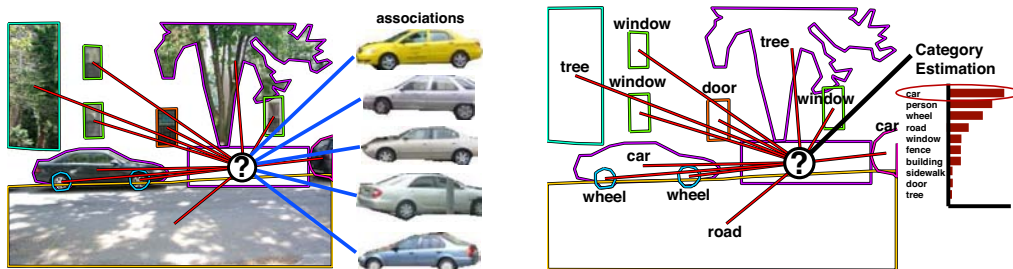

Figure 2: Torralba's Context Challenge: "How far can you go without running a local object detector?" The task is to reason about the identity of the hidden object (denoted by a "?") without local information. In our category-free Visual Memex model, object predictions are generated in the form of exemplar associations for the hidden object. In a category-based model, the category of the hidden object is directly estimated.

space. We create a similarity edge between two exemplars if they are deemed similar by each others' distance functions. We use a fixed $\lambda = .00001$ and $\sigma = 100$ for all exemplars.

## 3.2  Context Edges

When two objects occur inside a single image, we encode their 2-D spatial relationship into a context feature vector $\mathbf{f} \in \Re^{10}$ (visualized as red dotted edges in Figure 1). The context feature vector encodes relative overlap, relative displacement, relative scale, and relative height of the bottom-most pixel between two exemplar regions in a single image. This feature captures the spatial relationship between two regions and *does not take into account any appearance information* – it is a generalization of the spatial features used in [8]. We measure the similarity between two context features using a Gaussian kernel: $K(\mathbf{f}, \mathbf{f}') = e^{-\alpha_1 ||\mathbf{f} - \mathbf{f}'||^2}$ with $\alpha_1 = 1.0$.

## 3.3  Building the Visual Memex

We extract a large database of exemplar objects and their ground-truth segmentation masks from the LabelMe [22] dataset and learn the structure of the Visual Memex in an offline setting. We use objects from the 30 most frequently occurring categories in LabelMe. Similarity edges are created using the per-exemplar distance function learning framework of [19], and context edges are created each time two exemplars are observed in the same image. We have a total of $N = 87,802$ exemplars in the Visual Memex, $|E_S| = 276,782$ similarity edges, and $|E_C| = 989,106$ context edges.

## 4  Evaluating on the *Context Challenge*

The intuition that we would like to evaluate is that many useful regularities of the visual world are lost when dealing solely with categories (e.g. the side view of a building should associate more with a side view of a car than a frontal view of a car). The key motivation behind the Visual Memex is that context should depend on the appearance of an object and not just the category it belongs to. In order to test this hypothesis against the commonly held practice of abstracting away appearance into categories, we need a rich evaluation dataset as well as a meaningful evaluation task.

We found that the *Context Challenge* [11] recently proposed by Antonio Torralba fits our needs perfectly. The evaluation task is inspired by the question: "How far can you go without running an object detector?" The goal is to recognize a single object in the image without peeking at pixels belonging to that object. Torralba presented an algorithm for predicting the category and scale of an object using only contextual information [23], but his notion of context is *scene-centered* (where the appearance of the entire image is used for prediction). Since the context we wish study in this paper is object-centered, we use an object-centered formulation of the Context Challenge. While it is not clear if the absolute performance numbers on the Context Challenge are very meaningful in themselves, we feel that it is an ideal task for studying object-centered context and the role of categorization assumptions in such models.

In our variant of the Context Challenge, the goal is to predict the category of a hidden object $y_i$ solely based on its spatial relationships to some provided objects – without using the pixels belonging to the hidden object at all. For our study, we use manually provided regions and category labels of $K$ supporting objects inside a single image. We refer to the identities of the $K$ supporting objects in the image as $\{y_1, \ldots, y_K\}$ (where $y \in \{1, \ldots, |C|\}$) and the set of $K$ 2D spatial relationship features between each supporting object and the hidden object as $\{\mathbf{f}_{i1}, \ldots, \mathbf{f}_{iK}\}$.

## 4.1 Inference in the Visual Memex Model

In this section, we explain how to use the Visual Memex graph (automatically constructed from data) to perform inference for the Context Challenge hidden-object prediction task. Not making the "category assumption," the model is defined with respect to exemplar associations for the hidden object. Inference in the model returns a compatibility score between every exemplar and the hidden object, and can be though of as returning an ordered list of exemplar associations. Due to the nature of exemplar associations as opposed to category assignments, a supporting object can be associated with *multiple* exemplars as opposed to a *single* category. We create soft exemplar associations between each of the supporting objects and the exemplars in the Visual Memex using the similarity functions $\{D\}$ (see Section 3.1).

$\{S_1, \ldots, S_K\}$ are the appearance features for the $K$ supporting objects. $A_j^a$ is the affinity between exemplar $a$ in the Visual Memex and the $j$-th supporting object and is created by evaluating $S_j$ under $a$'s distance function $A_j^a = e^{-D_a(S_j)}$. $\Psi(e_i, e_j, \mathbf{f}_{ij})$ is the pairwise compatibility between exemplar $e_i$ and $e_j$ under the spatial feature $\mathbf{f}_{ij}$. Let $W_{ab}$ be the adjacency matrix representation of the similarity edges ($W_{uv} = [(u, v) \in E_S]$). Inference in the Visual Memex Model is done by optimizing the following conditional distribution which scores the assignment of an arbitrary exemplar $e_i$ to the hidden object based on contextual relations:

$$p(e_i | A_1, \ldots, A_K, \mathbf{f}_{i1}, \ldots, \mathbf{f}_{iK}) \quad \propto \quad \prod_{j=1}^{K} \sum_{a=1}^{N} A_j^a \Psi(e_i, e_a, \mathbf{f}_{ij}) \tag{2}$$

$$\log \Psi(e_i, e_j, \mathbf{f}_{ij}) \quad = \quad \frac{\sum_{(u,v) \in E_C} W_{iu} W_{jv} K(\mathbf{f}_{ij}, \mathbf{f}_{uv})}{\sum_{(u,v) \in E_C} W_{iu} W_{jv}} \tag{3}$$

The reason for the summation inside Equation 3 is that it aggregates contextual interactions from similar exemplars. By doing this, we effectively "densify" the contextual interactions in the Visual Memex. An interpretation of this densification procedure is that we are creating a kernel density estimator for an arbitrary pair of exemplars $(e_i, e_j)$ via a weighted sum of kernels placed at context features in the data set $\{\mathbf{f}_{uv}\} : (u, v) \in E_C$ where the weights $W_{iu} W_{jv}$ measure visual similarity between pairs $(e_i, e_j)$ and $(e_u, e_v)$.

We experimented with using a single kernel, $\Psi(e_i, e_j | \mathbf{f}_{ij}) = K(\mathbf{f}_{ij}, \mathbf{f}_{e_i, e_j})$, and found that the integration of multiple features via the densification described above is a key ingredient for successful Visual Memex inference.

Finally, after performing inference in the Visual Memex Model, we are left with a score for each exemplar. At this stage, as far as our model is concerned, the recognition has already been performed. However, since the task we are evaluated on is category-based, we combine the returned exemplars into a vote for categories using Luce's Axiom of Choice [17] which averages the exemplar responses per-category.

## 4.2 CoLA-based Parametric Model

We would like to evaluate the Visual Memex model against a more traditional, category-based framework with parametric inter-category relationships. One of the most recent and successful approaches is the CoLA model [8]. CoLA learns a set of parameters for each pair of categories which correspond to relative strengths of the four different top,above,below,inside spatial relationships. In the case of dealing with categories directly we consider a conditional distribution over the category of the hidden object $y_i$ that factors as a star graph with $K$ leaves (with the hidden object being connected to

all the supporting objects). $\boldsymbol{\theta}$ are model parameters, $\Psi$ is a pairwise potential that measures the compatibility of two categories with a specified spatial relationship, and $Z$ is a normalization constant such that the conditional distribution sums to 1.

$$p(y_i|y_1,\ldots,y_K,\mathbf{f}_{i1},\ldots,\mathbf{f}_{iK},\boldsymbol{\theta}) = \frac{1}{Z}\prod_{j=1}^{K}\Psi(y_i,y_j,\mathbf{f}_{ij},\boldsymbol{\theta}) \tag{4}$$

Following [8], we use a feature function $h(\mathbf{f})$ that computes the affinity between feature $\mathbf{f}$ and a set of prototypical spatial relationships. We automatically find $P$ prototypical spatial relationships by clustering all spatial feature vectors $\{\mathbf{f}\}$ in the training set via the popular Kmeans algorithm. Let $h(\mathbf{f}) \in \Re^P$ be the normalized vector of affinities to cluster centers $\{\mathbf{c}_1,\ldots,\mathbf{c}_P\}$. $\boldsymbol{\theta}$ is the set of all parameters in this model, with $\boldsymbol{\theta}(y_i,y_j) \in \Re^P$ being the parameters associated with the pair of categories $(y_i,y_j)$.

$$\log\Psi(y_i,y_j,\mathbf{f}_{ij},\boldsymbol{\theta}) = [h(\mathbf{f}_{ij})^T]\,\boldsymbol{\theta}(y_i,y_j) \tag{5}$$
$$h_i(\mathbf{f}) \propto e^{-\alpha||\mathbf{f}-\mathbf{c}_i||^2} \tag{6}$$

We tried using the four prototypical relationships corresponding to above, below, inside, and outside as in [8], but found that using Kmeans with significantly larger number of prototypes $P = 30$ produced superior results. For learning $\boldsymbol{\theta}$, we found the maximum likelihood $\boldsymbol{\theta}$ using gradient descent. The training objective function was optimized to mimic what happens during testing on the Context Challenge task. Since the distributions for the Context Challenge task are defined with respect to a single category variable (see Equation 4), we could compute the partition function directly and didn't require any approximations as in [8] (which required training in a loopy graph).

## 4.3   Reduced KDE Memex Model

Since the Visual Memex Model and the CoLA-inspired model make different assumptions with respect to objects (category-based vs. exemplar-based) and context (parametric vs. nonparametric), we feel it would also be useful to examine a hybrid model – dubbed the Reduced KDE Memex Model – which uses a nonparametric model of context but operates on object categories. The Reduced KDE Memex Model is created by collapsing all exemplars belonging to a single category into fully-connected components which can be thought of as adding categories into the Visual Memex graph. Identities between individual exemplars are lost, and thus we lose the fine details of a spatial context. By forming categories, we can no longer say a particular spatial relationship is between a blue side view of a car and an oblique brick building, we can only say it is a relationship between a car and a building. Now that we are left with an unordered bag of spatial relationships $\{\mathbf{f}\}$ between two categories, we need a way to measure compatibility between a newly observed $\mathbf{f}$ and the stored relationships.

We use the same form of the Context Challenge conditional distribution as in Equation 4. We use a Kernel Density Estimator(KDE) for every pair of categories, and the potential $\Psi$ can be thought of as a matrix of such estimators. The use of nonparametric potentials in graphical models has been already explored in the domain of texture analysis [24]. $\delta_{ij}$ is the Kronecker delta function.

$$\log\Psi(y_i,y_j,\mathbf{f}_{ij}) = \frac{\sum_{(u,v)\in E_C}\delta_{y_iy_u}\delta_{y_jy_v}K(\mathbf{f}_{ij},\mathbf{f}_{uv})}{\sum_{(u,v)\in E_C}\delta_{y_iy_u}\delta_{y_jy_v}} \tag{7}$$

The Reduced Memex model, being category-based and nonparametric, aggregates the spatial relationships across many different pairs of exemplars from two categories. While we used a fixed kernel $K$ which measures distance isotropically across the dimensions of $\mathbf{f}$, the advantage of such a nonparametric approach is that with enough data the particularities of $K$ do not matter. We also experimented with a Nearest Neighbor based model, but found the Kernel Density Estimation approach to be superior.

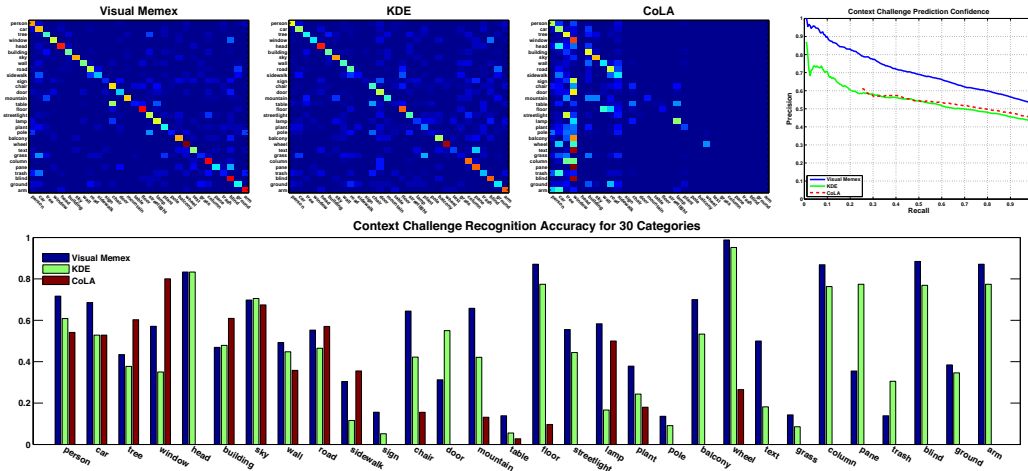

Figure 3: a.) Context Challenge confusion matrices for the 3 methods: Visual Memex, KDE, and CoLA. b.) Recognition Precision versus Recall when thresholding output based on confidence. c) Side by side comparison of the 3 methods' accuracies for 30 categories.

## 5   Results and Discussion

For the Context Challenge evaluation, we use 200 randomly selected densely labeled images from LabelMe [22]. Our testset contains 3048 total objects from 30 different categories. For an image with $K$ objects, we solve $K$ Context Challenge problems with one hidden object and $K$-1 supporting objects. Qualitative results on this prediction task can be seen in Figure 4.

We evaluate the performance of our Visual Memex model, the Reduced Memex KDE model, and the CoLA-inspired model with respect to categorization performance (confusion matrices can be seen in top left of Figure 3). The overall recognition accuracy of the Visual Memex Model, Reduced Memex Model, and CoLA are .527, .430, and .457 respectively. Note that the Visual Memex Model performs significantly better than the baselines. Taking a closer look at the per-category accuracies of the three methods (see bottom of Figure 3), we see that the CoLA-based method fails on many categories. The average per-category recognition accuracies of the three methods are: .534, .454, and .213. The Visual Memex Model still performs the best, but we see a significant drop in performance for the category-based CoLA method. CoLA is biased towards the popular categories, returning the most frequently occurring category (window) quite often. Overall, the Visual Memex Model achieves the best performance for 21 out of the 30 categories.

In addition, we plot precision recall curves for each of the three methods to determine if high confidence returned by each model is correlated with high recognition rates (top right of Figure 3). The Visual Memex model has the most significant high-precision low-recall regime, suggesting that its confidence is a good measure of success. The relatively flat curve for the CoLA method is related to the problem of overcompensation for popular classes as mentioned above. The distributions returned by CoLA tend to degenerate to a single non-zero value (most often on one of the popular categories such as window). This is why the maximum probability returned by CoLA isn't a good measure of confidence.

We also demonstrate the power of the Visual Memex to predict *appearance* solely based on contextual interactions with other objects and their visual appearance. The middle row of Figure 4 demonstrates some of these associations. Note how in row 1, a plausible viewpoint is selected rather than just a random car. In row 3 we see that the appearance of snow on one mountain suggests that the other portion of the image also contains a snowy mountain. In summary, we presented a category-free Visual Memex Model and applied it to the task of contextual object recognition within the experimental framework of the *Context Challenge*. Our experiments confirm our intuition that moving beyond categories is beneficial for improved modeling of relationships between objects.

**Acknowledgements.** This research was in part funded by NSF CAREER award IIS-0546547, NSF Graduate Research Fellowship, a Guggenheim Fellowship, as well as generous gift from Google. A. Efros thanks the WILLOW team at ENS, Paris for their hospitality.

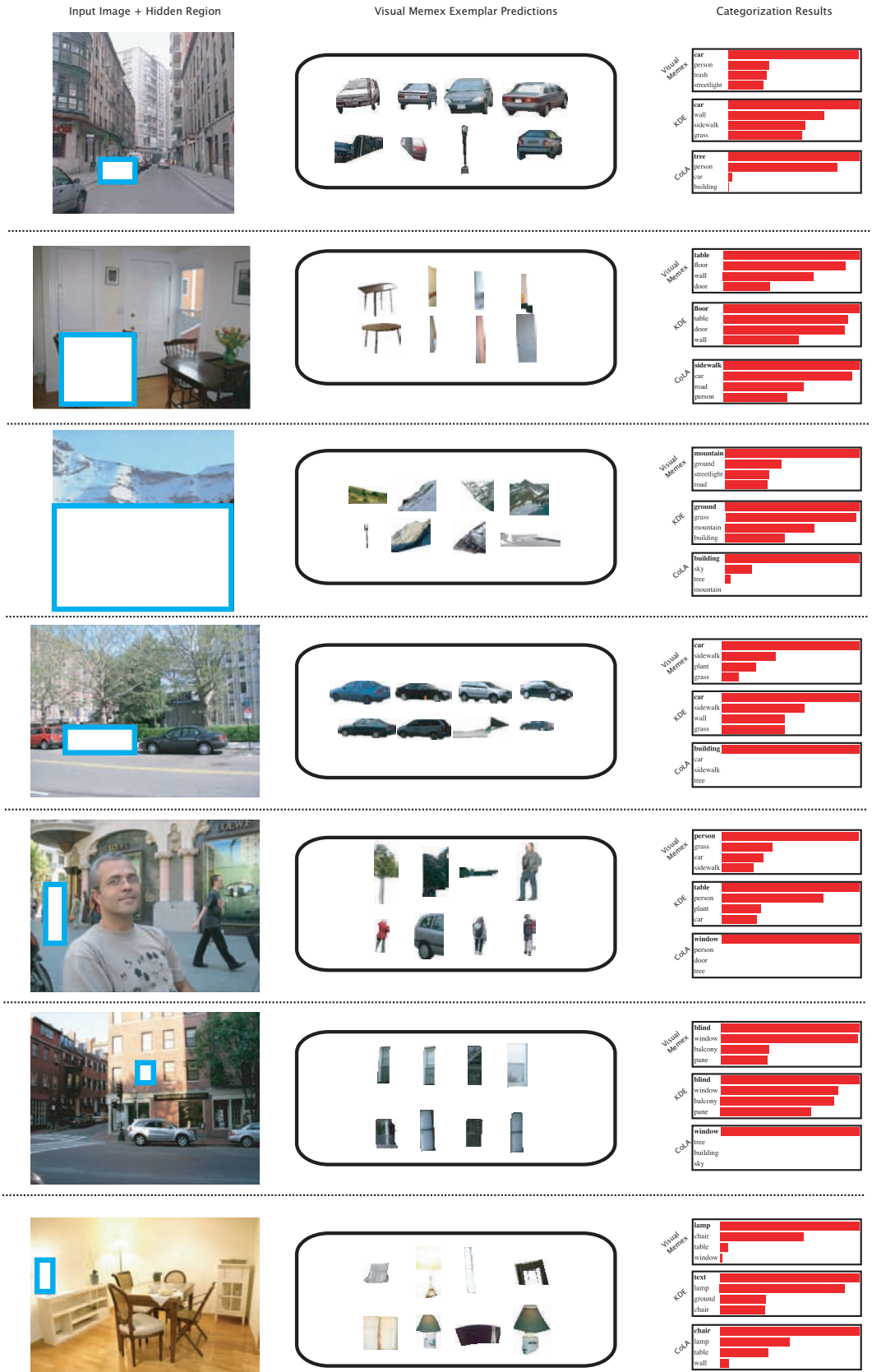

Figure 4: Qualitative Results on the Context Challenge. Exemplar predictions are from the Visual Memex model and categorization results are from the Visual Memex model, the KDE Model, and CoLA[8].

# References

[1] Moshe Bar and Shimon Ullman. Spatial context in recognition. *Perception*, 25:343–352, 1996. 1

[2] A.R. Hanson and E.M. Riseman. Visions: A computer system for interpreting scenes. *Computer Vision Systems*, pages 303–333, 1978. 1

[3] T.M. Strat and M.A. Fischler. Context-based vision: Recognizing objects using information from both 2-d and 3-d imagery. *PAMI*, 13:1050–1065, 1991. 1

[4] Xuming He, Richard S. Zemel, and Miguel Á. Carreira-Perpiñán. Multiscale conditional random fields for image labeling. *CVPR*, pages 695–702, 2004. 1

[5] Sanjiv Kumar and Martial Hebert. A hierarchical field framework for unified context-based classification. *ICCV*, 2005. 1

[6] Jamie Shotton, John M. Winn, Carsten Rother, and Antonio Criminisi. Textonboost: Joint appearance, shape and context modeling for multi-class object recognition and segmentation. *ECCV*, 2006. 1

[7] Andrew Rabinovich, Anrea Vedaldi, Carolina Galleguillos, Eric Wiewora, and Serge Belongie. Objects in context. *ICCV*, 2007. 1

[8] Carolina Galleguillos, Andrew Rabinovich, and Serge Belongie. Object categorization using co-occurrence, location and appearance. *ECCV*, 2008. 1, 4, 5, 6, 8

[9] Devi Parikh, C. Lawrence Zitnick, and Tsuhan Chen. From appearance to context-based recognition: Dense labeling in small images. *CVPR*, 2008. 1

[10] Bryan C. Russell, Antonio Torralba, Ce Liu, Rob Fergus, and William T. Freeman. Object recognition by scene alignment. *NIPS*, 2007. 1

[11] Antonio Torralba. The context challenge. *http://web.mit.edu/torralba/www/carsAndFacesInContext.html*. 1, 4

[12] Aristotle. *Categories*. 2

[13] Ludwig Wittgenstein. *Philosophical Investigations*. Blackwell Publishing, 1953. 2

[14] Eleanor Rosch. Principles of categorization. *Cognition and Categorization*, pages 27–48, 1978. 2

[15] Shimon Edelman. Representation, similarity and the chorus of prototypes. *Minds and Machines*, 1995. 2

[16] Ariadna Quattoni, M. Collins, and Trevor Darrell. Transfer learning for image classification with sparse prototype representations. *CVPR*, 2008. 2

[17] D. L. Medin and M.M. Schaffer. Context theory of classification learning. *Psychological Review*, 85:207–238, 1978. 2, 5

[18] Robert M. Nosofsky. Attention, similarity, and the identification-categorization relationship. *Journal of Experimental Psychology: General*, 115(1):39–57, 1986. 2

[19] Tomasz Malisiewicz and Alexei A. Efros. Recognition by association via learning per-exemplar distances. *CVPR*, 2008. 2, 3, 4

[20] Moshe Bar. The proactive brain: memory for predictions. *Philosophical Transactions of the Royal Society B*, 364:1235–1243, 2009. 2

[21] Vannevar Bush. As we may think. *The Atlantic Monthly*, 1945. 2

[22] Bryan Russell, Antonio Torralba, Kevin Murphy, , and William T. Freeman. Labelme: a database and web-based tool for image annotation. *International Journal of Computer Vision*, 77:157–173, 2008. 4, 7

[23] Antonio Torralba. Contextual priming for object detection. *International Journal of Computer Vision*, 53:169–191, 2003. 4

[24] Rupert Paget and I. D. Longstaff. Texture synthesis via a noncausal nonparametric multiscale markov random field. *IEEE Transactions on Image Processing*, 1998. 6

